# Learning to Predict Visibility and Invisibility from Occlusion Events

Jonathan A. Marshall     Richard K. Alley

Robert S. Hubbard

Department of Computer Science, CB 3175, Sitterson Hall
University of North Carolina, Chapel Hill, NC 27599-3175, U.S.A.

marshall@cs.unc.edu, 919-962-1887, fax 919-962-1799

## Abstract

Visual occlusion events constitute a major source of depth information. This paper presents a self-organizing neural network that learns to detect, represent, and predict the visibility and invisibility relationships that arise during occlusion events, after a period of exposure to motion sequences containing occlusion and disocclusion events. The network develops two parallel opponent channels or "chains" of lateral excitatory connections for every resolvable motion trajectory. One channel, the "On" chain or "visible" chain, is activated when a moving stimulus is visible. The other channel, the "Off" chain or "invisible" chain, carries a persistent, *amodal* representation that predicts the motion of a formerly visible stimulus that becomes invisible due to occlusion. The learning rule uses disinhibition from the On chain to trigger learning in the Off chain. The On and Off chain neurons can learn *separate* associations with object depth ordering. The results are closely related to the recent discovery (Assad & Maunsell, 1995) of neurons in macaque monkey posterior parietal cortex that respond selectively to inferred motion of invisible stimuli.

## 1   INTRODUCTION: LEARNING ABOUT OCCLUSION EVENTS

Visual occlusion events constitute a major source of depth information. Yet little is known about the neural mechanisms by which visual systems use occlusion events to infer the depth relations among visual objects. What is the structure of such mechanisms? Some possible answers to this question are revealed through an analysis of learning rules that can cause such mechanisms to self-organize.

Evidence from psychophysics (Kaplan, 1969; Nakayama & Shimojo, 1992; Nakayama, Shimojo, & Silverman, 1989; Shimojo, Silverman, & Nakayama, 1988, 1989; Yonas, Craton, & Thompson, 1987) and neurophysiology (Assad & Maunsell, 1995; Frost, 1993) suggests that the process of determining relative depth from occlusion events operates at an early stage of visual processing. Marshall (1991) describes evidence that suggests that the same early processing mechanisms maintain a representation of temporarily occluded objects for some amount

of time after they have disappeared behind an occluder, and that these representations of invisible objects interact with other object representations, in much the same manner as do representations of visible objects. The evidence includes the phenomena of kinetic subjective contours (Kellman & Cohen, 1984), motion viewed through a slit (Parks' Camel) (Parks, 1965), illusory occlusion (Ramachandran, Inada, & Kiama, 1986), and interocular occlusion sequencing (Shimojo, Silverman, & Nakayama, 1988).

## 2   PERCEPTION OF OCCLUSION AND DISOCCLUSION EVENTS: AN ANALYSIS

The neural network model exploits the visual changes that occur at occlusion boundaries to form a mechanism for detecting and representing object visibility/invisibility information. The set of learning rules used in this model is an extended version of one that has been used before to describe the formation of neural mechanisms for a variety of other visual processing functions (Hubbard & Marshall, 1994; Marshall, 1989, 1990ac, 1991, 1992; Martin & Marshall, 1993).

Our analysis is derived from the following visual *predictivity principle*, which may be postulated as a fundamental principle of neural organization in visual systems: *Visual systems represent the world in terms of predictions of its appearance, and they reorganize themselves to generate better predictions.* To maximize the correctness and completeness of its predictions, a visual system would need to predict the motions and visibility/invisibility of all objects in a scene. Among other things, it would need to predict the disappearance of an object moving behind an occluder and the reappearance of an object emerging from behind an occluder.

A consequence of this postulate is that occluded objects must, at some level, continue to be represented even though they are invisible. Moreover, the representation of an object must distinguish whether the object is visible or invisible; otherwise, the visual system could not determine whether its representations predict visibility or invisibility, which would contravene the predictivity principle. Thus, simple single-channel prediction schemes like the one described by Marshall (1989, 1990a) are inadequate to represent occlusion and disocclusion events.

## 3   A MODEL FOR GROUNDED LEARNING TO PREDICT VISIBILITY AND INVISIBILITY

The initial structure of the Visible/Invisible network model is given in Figure 1A. The network self-organizes in response to a training regime containing many input sequences representing motion with and without occlusion and disocclusion events. After a period of self-organization, the specific connections that a neuron receives (Figure 1B) determine whether it responds to visible or invisible objects. A neuron that responds to visible objects would have strong bottom-up input connections, and it would also have strong time-delayed lateral excitatory input connections. A neuron that responds selectively to invisible objects would *not* have strong bottom-up connections, but it would have strong lateral excitatory input connections. These lateral inputs would transmit to the neuron evidence that a previously visible object existed. The neurons that respond to invisible objects must operate in a way that allows lateral input excitation alone to activate the neurons supraliminally, in the absence of bottom-up input excitation from actual visible objects.

## 4   SIMULATION OF A SIMPLIFIED NETWORK

### 4.1   INITIAL NETWORK STRUCTURE

The simulated network, shown in Figure 2, describes a simplified one-dimensional subnetwork (Marshall & Alley, 1993) of the more general two-dimensional network. Layer 1 is restricted to a set of motion-sensitive neurons corresponding to one rightward motion trajectory.

The L+ connections in the simulation have a signal transmission latency of one time unit. Restricting the lateral connections to a single time delay and to a single direction limits the simulation to representing a single speed and direction of motion; these results are therefore preliminary. This restriction reduced the number of connections and made the simulation much faster.

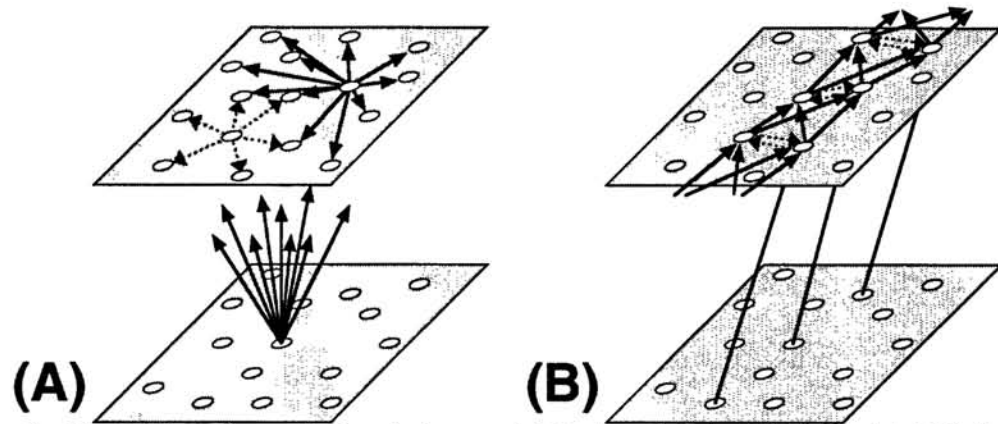

Figure 1: Model of a self-organized occlusion-event detector network. (A) Network is initially organized nonspecifically, so that each neuron receives roughly homogeneous input connections: feedforward, bottom-up excitatory ("B+") connections from a preprocessing stage of motion-tuned neurons (bottom-up solid arrows), lateral inhibitory ("L−") connections (dotted arrows), and time-delayed lateral excitatory ("L+") connections (lateral solid arrows). (B) After exposure during a developmental period to many motion sequences containing occlusion and disocclusion events, the network learns a highly specific connection structure. The previously homogeneous network bifurcates into two parallel opponent channels for every resolvable motion trajectory: some neurons keep their bottom-up connections and others lose them. The channels for one trajectory are shown. Neurons from the two opponent channels are strongly linked by lateral inhibitory connections (dotted arrows). Time-delayed lateral excitatory connections cause stimulus information (priming excitation, or "prediction signals") to propagate along the channels.

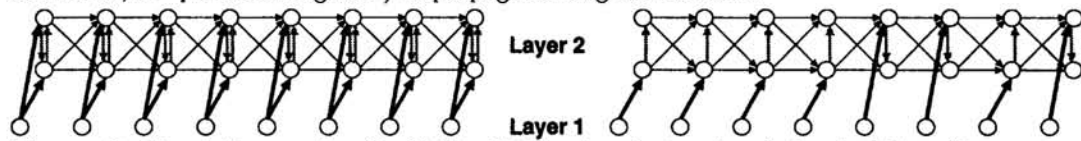

Figure 2: Simulation results. (Left) Simulated network structure before training. Neurons are wired homogeneously from the input layer. (Right) After training, some of the neurons lose their bottom-up input connections.

## 4.2    USING DISINHIBITION TO CONTROL THE LEARNING OF OCCLUSION RELATIONS

This paper describes one method for learning occlusion relations. Other methods may also work. The method involves extending the EXIN (excitatory+inhibitory) learning scheme described by Marshall (1992, 1995). The EXIN scheme uses a variant of a Hebb rule to govern learning in the bottom-up and time-delayed lateral excitatory connections, plus an anti-Hebb rule to govern learning in the lateral inhibitory connections.

The EXIN system was extended by letting inhibitory connections exert a *disinhibitory* effect under certain regulated conditions. The disinhibition rule was chosen because it constitutes a simple way that the unexpected failure of a neuron to become activated (e.g., when an object disappears behind an occluder) can cause some *other* neuron to become activated. That other neuron can then learn, becoming selective for invisible object motion. Thus, the representations of visible objects are protected from losing their bottom-up input connections during occlusion events.

In this way, the network can learn *separate* representations for visible and invisible stimuli. The representations of invisible objects are allowed to develop only to the extent that the neurons representing visible objects explicitly disclaim the "right" to represent the objects. These properties prevent the network from losing complete grounded contact with actual bottom-up visual input, while at the same time allowing *some* neurons to lose their direct bottom-up input connections.

The disinhibition produces an excitatory response at the target neurons. Disinhibition is generated according to the following rule: *When a neuron has strong,*

*active lateral excitatory input connections and strong but inactive bottom-up input connections, then it tends to disinhibit the neurons to which it projects inhibitory connections.* This implements a type of differencing operation between lateral and bottom-up excitation. Because the disinhibition tends to excite the recipient neurons, it causes one (or possibly more) of the recipient neurons to become active and thereby enables that neuron to learn.

The lateral excitation that a neuron receives can be viewed as a prediction of the neuron's activation. If that prediction is not matched by actual bottom-up excitation, then a *shortfall* (prediction failure) has occurred, probably indicating an occlusion event.

Each neuron's disinhibition input was combined with its bottom-up excitatory input and its lateral excitatory input to form a total excitatory input signal. Either bottom-up excitation or disinhibition alone could contribute toward a neuron's excitation. However, lateral excitation could merely amplify the other signals and could not alone excite the neuron. This prevented neurons from learning in response to lateral excitation alone.

## 4.3   DISINHIBITION LETS THE NETWORK LEARN TO RESPOND TO INVISIBLE OBJECTS

During continuous motion sequences, without occlusion or disocclusion, the system operates similarly to a system with the standard EXIN learning rules (Marshall, 1990b, 1995): lateral excitatory "chains" of connections are learned across sequences of neurons along a motion trajectory. Marshall (1990a) showed that such chains form in 2-D networks with multiple speeds and multiple directions of motion.

During occlusion events, some predictive lateral excitatory signals reach neurons that have strong but inactive bottom-up excitatory connections. The neurons reached by this excitation pattern disinhibit, rather than inhibit, their competitor neurons. Over the course of many occlusion events, such neurons become increasingly selective for the inferred motion of an invisible object: their bottom-up input connections weaken, and their lateral inhibitory input connections strengthen.

More than one neuron receives L+ signals after every neuron activation; the recipients of each neuron's L+ output connections represent the (learned) possible sequents of the neuron's activation. But at most one of those sequents actually receives both B+ and L+ signals: the one that corresponds to the actual stimulus. This winner neuron receives the disinhibition from the other neurons receiving L+ excitation; its competitive advantage over the other neurons is thus reinforced.

## 4.4   SIMULATION TRAINING

The sequences of input training data consisted of a single visual feature moving with constant velocity across the 1-D visual field. When this stimulus was visible, its presence was indicated by strong activation of an input neuron in Layer 1. While occluded, the stimulus would produce no activation in Layer 1. The stimulus occasionally disappeared "behind" an occluder and reappeared at a later time and spatial position farther along the same trajectory. After some duration, the stimulus was removed and replaced by a new stimulus. The starting positions and lifetimes of the stimuli and occluders were varied randomly within a fixed range.

The network was trained for 25,000 input pattern presentations. The stability of the connection weights was verified by additional training for 50,000 presentations.

## 4.5   SIMULATION RESULTS: ARCHITECTURE

The second stage of neurons gradually underwent a self-organized *bifurcation* into two distinct pools of neurons, as shown in Figure 2B. These pools consist of two parallel opponent channels or "chains" of lateral excitatory connections for every resolvable motion trajectory. One channel, the "On" chain or "visible" chain, was active when a moving stimulus became visible. The other channel, the "Off" chain or "invisible" chain, was active when a formerly visible stimulus became invisible. The model is thus named the *Visible/Invisible* model. The bifurcation may be analogous to the activity-dependent stratification of cat retinal ganglion cells into separate On and Off layers, described by Bodnarenko and Chalupa (1993).

## 4.6   SIMULATION RESULTS: OPERATION

The On chain carries a predictive *modal* representation of the visible stimulus. The Off chain carries a persistent, *amodal* representation that predicts the motion

of the invisible stimulus. The shading of the neurons in Figure 3 shows the neuron activations of the final, trained network simulation during an occlusion–disocclusion sequence. The following noteworthy behaviors were observed in the test.

- When the stimulus was visible, it was represented by activation in the On channel.
- When the stimulus became invisible, its representation was carried in the Off channel. The Off channel did not become active until the visible stimulus disappeared.
- The activations representing the visible stimulus became stronger (toward an asymptote) at successive spatial positions, because of the propagation of accumulating evidence for the presence of the stimulus (Martin & Marshall, 1993).
- The activation representing the invisible stimulus decayed at successive spatial positions. Thus, representations of invisible stimuli did not remain active indefinitely.
- When the stimulus reappeared (after a sufficiently brief occlusion), its activation in the On channel was greater than its initial activation in the On channel. Thus, the representation carried across the Off channel helps maintain the perceptual stability of the stimulus despite its being temporarily occluded along parts of its trajectory.

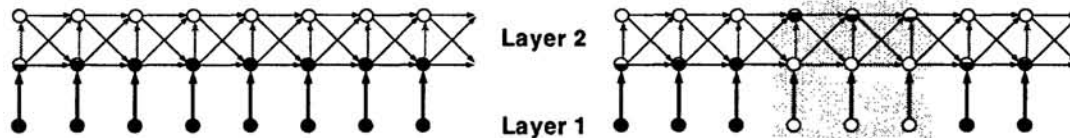

Figure 3: Simulated network operation after learning. The learning procedure causes the representation of each trajectory to split into two parallel opponent channels. The Visible and Invisible channel pair for a single trajectory are shown. The display has been arranged so that all the Visible channel neurons are on the same row (Layer 2, lower row); likewise the Invisible channel neurons (Layer 2, upper row). Solid arrows indicate excitatory connections. Gray arrows indicate lateral inhibitory connections. (Left) The network's responses to an unbroken rightward motion of the stimulus are shown. The activities of the network at successive moments in time have been combined into a single network display; each horizontal position in the figure represents a different moment in time as well as a different position in the network. The stimulus successively activates motion detectors (solid circles) in Layer 1. The activation of the responding neuron in the second layer builds toward an asymptote, reaching full activation by the fourth frame. (Right) The network's responses to a broken (occluded) rightward motion sequence are shown. When the stimulus reaches the region indicated by gray shading, it disappears behind a simulated occluder. The network responds by successively activating neurons in the Invisible channel. When the stimulus emerges from behind the occluder (end of gray shading), it is again represented by activation in the Visible channel.

## 5   DISCUSSION

### 5.1   PSYCHOPHYSICAL ISSUES AND PREDICTIONS

Several visual phenomena (Burr, 1980; Piaget, 1954; Shimojo, Silverman, & Nakayama, 1988) support the notion that early processing mechanisms maintain a dynamic representation of temporarily occluded objects for some amount of time after they disappear (Marshall, 1991). In general, the duration of such representations should vary as a function of many factors, including top-down cognitive expectations, stimulus complexity, and Gestalt grouping.

### 5.2   ALTERNATIVE MECHANISMS

Another model besides the Visible/Invisible model was studied extensively: a *Visible/Virtual* system, which would develop some neurons that respond to visible objects and others that respond to both visible and invisible objects (i.e., to "virtual" objects). There is a functional equivalence between such a Visible/Virtual system and a Visible/Invisible system: the same information about visibility and invisibility can be determined by examining the activations of the neurons. Activity in a Virtual channel neuron, paired with inactivity in a corresponding Visible channel neuron, would indicate the presence of an invisible stimulus.

## 5.3   NEUROPHYSIOLOGICAL CORRELATES

Assad and Maunsell (1995) recently described their remarkable discovery of neurons in macaque monkey posterior parietal cortex that respond selectively to the inferred motion of invisible stimuli. This type of neuron responded more strongly to the disappearance and reappearance of a stimulus in a task where the stimulus' "inferred" trajectory would pass through the neuron's receptive field than in a task where the stimulus would disappear and reappear in the same position. Most of these neurons also had a strong off-response, which in the present models is closely correlated with inferred motion. Thus, the results of Assad and Maunsell (1995) are more directly consistent with the Visible/Virtual model than with the Visible/Invisible model. Although this paper describes only one of these models, both models merit investigation.

## 5.4   LEARNING ASSOCIATIONS BETWEEN VISIBILITY AND RELATIVE DEPTH

The activation of neurons in the Off channels is highly correlated with the activation of other neurons elsewhere in the visual system, specifically neurons whose activation indicates the presence of other objects acting as occluders. Simple associative Hebb-type learning lets such occluder-indicator neurons and the Off channel neurons gradually establish reciprocal excitatory connections to each other.

After such reciprocal excitatory connections have been learned, activation of occluder-indicator neurons at a given spatial position causes the network to favor the Off channel in its predictions – i.e., to predict that a moving object will be invisible at that position. Thus, the network learns to use occlusion information to generate better predictions of the visibility/invisibility of objects.

Conversely, the activation of Off channel neurons causes the occluder-indicator neurons to receive excitation. The disappearance of an object excites the representation of an occluder at that location. If the representation of the occluder was not previously activated, then the excitation from the Off channel may even be strong enough to activate it alone. Thus, disappearance of moving visual objects constitutes evidence for the presence of an inferred occluder. These results will be described in a later paper.

## 5.5   LIMITATIONS AND FUTURE WORK

The Visible/Invisible model presented in this paper describes some of the processes that may be involved in detecting and representing depth from occlusion events. There are other major issues that have not been addressed in this paper. For example, how can the system handle real 2-D or 3-D objects, composed of many visual features grouped together across space, instead of mere point stimuli? How can it handle partial occlusion of objects? How can it handle nonlinear trajectories? How exactly can the associative links between occluding and occluded objects be formed? How can it handle transparency?

# 6   CONCLUSIONS

Perception of relative depth from occlusion events is a powerful, useful, but poorly-understood capability of human and animal visual systems. We have presented an analysis based on predictivity: a visual system that can predict the visibility/invisibility of objects during occlusion events possesses (*ipso facto*) a good representation of relative depth. The analysis implies that the representations for visible and invisible objects must be distinguishable. We have implemented a model system in which distinct representations for visible and invisible features self-organize in response to exposure to motion sequences containing simulated occlusion and disocclusion events. When a moving feature fails to appear approximately where and when it is predicted to appear, the mismatch between prediction and the actual image triggers an unsupervised learning rule. Over many motions, the learning leads to a bifurcation of a network layer into two parallel opponent channels of neurons. Prediction signals in the network are carried along motion trajectories by specific chains of lateral excitatory connections. These chains also cause the representation of invisible features to propagate for a limited time along the features' trajectories. The network uses shortfall (differencing) and disinhibition to maintain grounding of the representations of invisible features.

## Acknowledgements
Supported in part by ONR (N00014-93-1-0208), NEI (EY09669), a UNC-CH Junior Faculty Development Award, an ORAU Junior Faculty Enhancement Award from Oak Ridge Associated Universities, the Univ. of Minnesota Center for Research in Learning, Perception, and Cognition, NICHHD (HD-07151), and the Minnesota Supercomputer Institute.

We thank Kevin Martin, Stephen Aylward, Eliza Graves, Albert Nigrin, Vinay Gupta, George Kalarickal, Charles Schmitt, Viswanath Srikanth, David Van Essen, Christof Koch, and Ennio Mingolla for valuable discussions.

## References

Assad JA, Maunsell JHR (1995) Neuronal correlates of inferred motion in macaque posterior parietal cortex. *Nature* 373:518–521.

Bodnarenko SR, Chalupa LM (1993) Stratification of On and Off ganglion cell dendrites depends on glutamate-mediated afferent activity in the developing retina. *Nature* 364:144–146.

Burr D (1980) Motion smear. *Nature* 284:164–165.

Frost BJ (1993) Subcortical analysis of visual motion: Relative motion, figure-ground discrimination and induced optic flow. *Visual Motion and Its Role in the Stabilization of Gaze*, Miles FA, Wallman J (Eds). Amsterdam: Elsevier Science, 159–175.

Hubbard RS, Marshall JA (1994) Self-organizing neural network model of the visual inertia phenomenon in motion perception. Technical Report 94-001, Department of Computer Science, University of North Carolina at Chapel Hill. 26 pp.

Kaplan GA (1969) Kinetic disruption of optical texture: The perception of depth at an edge. *Perception & Psychophysics* 6:193–198.

Kellman PJ, Cohen MH (1984) Kinetic subjective contours. *Perception & Psychophysics* 35:237–244.

Marshall JA (1989) Self-organizing neural network architectures for computing visual depth from motion parallax. *Proceedings of the International Joint Conference on Neural Networks*, Washington DC, II:227–234.

Marshall JA (1990a) Self-organizing neural networks for perception of visual motion. *Neural Networks* 3:45–74.

Marshall JA (1990b) A self-organizing scale-sensitive neural network. *Proceedings of the International Joint Conference on Neural Networks*, San Diego, CA, III:649–654.

Marshall JA (1990c) Adaptive neural methods for multiplexing oriented edges. *Intelligent Robots and Computer Vision IX: Neural, Biological, and 3-D Methods*, Casasent DP (Ed), Proceedings of the SPIE 1382, Boston, MA, 282–291.

Marshall JA (1991) Challenges of vision theory: Self-organization of neural mechanisms for stable steering of object-grouping data in visual motion perception. *Stochastic and Neural Methods in Signal Processing, Image Processing, and Computer Vision*, Chen SS (Ed), Proceedings of the SPIE 1569, San Diego, CA, 200–215.

Marshall JA (1992) Unsupervised learning of contextual constraints in neural networks for simultaneous visual processing of multiple objects. *Neural and Stochastic Methods in Image and Signal Processing*, Chen SS (Ed), Proceedings of the SPIE 1766, San Diego, CA, 84–93.

Marshall JA (1995) Adaptive perceptual pattern recognition by self-organizing neural networks: Context, uncertainty, multiplicity, and scale. *Neural Networks* 8:335–362.

Marshall JA, Alley RK (1993) A self-organizing neural network that learns to detect and represent visual depth from occlusion events. *Proceedings of the AAAI Fall Symposium on Machine Learning and Computer Vision*, Bowyer K, Hall L (Eds), 70–74.

Martin KE, Marshall JA (1993) Unsmearing visual motion: Development of long-range horizontal intrinsic connections. *Advances in Neural Information Processing Systems, 5*, Hanson SJ, Cowan JD, Giles CL (Eds). San Mateo, CA: Morgan Kaufmann Publishers, 417–424.

Nakayama K, Shimojo S (1992) Experiencing and perceiving visual surfaces. *Science* 257:1357–1363.

Nakayama K, Shimojo S, Silverman GH (1989) Stereoscopic depth: Its relation to image segmentation, grouping, and the recognition of occluded objects. *Perception* 18:55–68.

Parks T (1965) Post-retinal visual storage *American Journal of Psychology* 78:145–147.

Piaget J (1954) *The Construction of Reality in the Child*. New York: Basic Books.

Ramachandran VS, Inada V, Kiama G (1986) Perception of illusory occlusion in apparent motion. *Vision Research* 26:1741–1749.

Shimojo S, Silverman GH, Nakayama K (1989) Occlusion and the solution to the aperture problem for motion. *Vision Research* 29:619–626.

Yonas A, Craton LG, Thompson WB (1987) Relative motion: Kinetic information for the order of depth at an edge. *Perception & Psychophysics* 41:53–59.